# A Theory of Retinal Population Coding

**Eizaburo Doi**
Center for the Neural Basis of Cognition
Carnegie Mellon University
Pittsburgh, PA 15213
edoi@cnbc.cmu.edu

**Michael S. Lewicki**
Center for the Neural Basis of Cognition
Carnegie Mellon University
Pittsburgh, PA 15213
lewicki@cnbc.cmu.edu

## Abstract

Efficient coding models predict that the optimal code for natural images is a population of oriented Gabor receptive fields. These results match response properties of neurons in primary visual cortex, but not those in the retina. Does the retina use an optimal code, and if so, what is it optimized for? Previous theories of retinal coding have assumed that the goal is to encode the maximal amount of information about the sensory signal. However, the image sampled by retinal photoreceptors is degraded both by the optics of the eye and by the photoreceptor noise. Therefore, de-blurring and de-noising of the retinal signal should be important aspects of retinal coding. Furthermore, the ideal retinal code should be robust to neural noise and make optimal use of all available neurons. Here we present a theoretical framework to derive codes that simultaneously satisfy all of these desiderata. When optimized for natural images, the model yields filters that show strong similarities to retinal ganglion cell (RGC) receptive fields. Importantly, the characteristics of receptive fields vary with retinal eccentricities where the optical blur and the number of RGCs are significantly different. The proposed model provides a unified account of retinal coding, and more generally, it may be viewed as an extension of the Wiener filter with an arbitrary number of noisy units.

## 1 Introduction

What are the computational goals of the retina? The retina has numerous specialized classes of retinal ganglion cells (RGCs) that are likely to subserve a variety of different tasks [1]. An important class directly subserving visual perception is the midget RGCs (mRGCs) which constitute 70% of RGCs with an even greater proportion at the fovea [1]. The problem that mRGCs face should be to maximally preserve signal information in spite of the limited representational capacity, which is imposed both by neural noise and the population size. This problem was recently addressed (although not specifically as a model of mRGCs) in [2], which derived the theoretically optimal linear coding method for a noisy neural population. This model is not appropriate, however, for the mRGCs, because it does not take into account the noise in the retinal image (Fig. 1). Before being projected on the retina, the visual stimulus is distorted by the optics of the eye in a manner that depends on eccentricity [3]. This retinal image is then sampled by cone photoreceptors whose sampling density also varies with eccentricity [1]. Finally, the sampled image is noisier in the dimmer illumination condition [4]. We conjecture that the computational goal of mRGCs is to represent the maximum amount of information about the underlying, non-degraded image signal subject to limited coding precision and neural population size.

Here we propose a theoretical model that achieves this goal. This may be viewed as a generalization of both Wiener filtering [5] and robust coding [2]. One significant characteristic of the proposed model is that it can make optimal use of an arbitrary number of neurons in order to preserve the maximum amount of signal information. This allows the model to predict theoretically optimal representations at any retinal eccentricity in contrast to the earlier studies [4, 6, 7, 8].

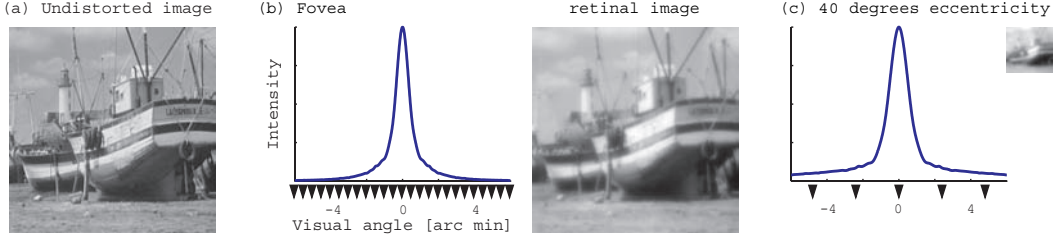

Figure 1: Simulation of retinal images at different retinal eccentricities. (a) Undistorted image signal. (b) The convolution kernel at the fovea [3] superimposed on the photoreceptor array indicated by triangles under the x-axis [1]. (c) The same as in (b) but at 40 degrees of retinal eccentricity.

## 2   The model

First let us define the problem (Fig. 2). We assume that data sampled by photoreceptors (referred to as the observation) $\mathbf{x} \in \mathbb{R}^N$ are blurred versions of the underlying image signal $\mathbf{s} \in \mathbb{R}^N$ with additive white noise $\boldsymbol{\nu} \sim \mathcal{N}(0, \sigma_\nu^2 \mathbf{I}_N)$,

$$\mathbf{x} = \mathbf{H}\mathbf{s} + \boldsymbol{\nu} \tag{1}$$

where $\mathbf{H} \in \mathbb{R}^{N \times N}$ implements the optical blur. To encode the image, we assume that the observation is linearly transformed into an $M$-dimensional representation. To model limited neural precision, it is assumed that the representation is subject to additive channel noise, $\boldsymbol{\delta} \sim \mathcal{N}(0, \sigma_\delta^2 \mathbf{I}_M)$. The noisy neural representation is therefore expressed as

$$\mathbf{r} = \mathbf{W}(\mathbf{H}\mathbf{s} + \boldsymbol{\nu}) + \boldsymbol{\delta} \tag{2}$$

where each row of $\mathbf{W} \in \mathbb{R}^{M \times N}$ corresponds to a receptive field. To evaluate the amount of signal information preserved in the representation, we consider a linear reconstruciton $\hat{\mathbf{s}} = \mathbf{A}\mathbf{r}$ where $\mathbf{A} \in \mathbb{R}^{N \times M}$. The residual is given by

$$\boldsymbol{\epsilon} = (\mathbf{I}_N - \mathbf{A}\mathbf{W}\mathbf{H})\mathbf{s} - \mathbf{A}\mathbf{W}\boldsymbol{\nu} - \mathbf{A}\boldsymbol{\delta}, \tag{3}$$

where $\mathbf{I}_N$ is the $N$-dimensional identity matrix, and the mean squared error (MSE) is

$$\mathcal{E} = \text{tr}[\boldsymbol{\Sigma}_\mathbf{s}] - 2\,\text{tr}[\mathbf{A}\mathbf{W}\mathbf{H}\boldsymbol{\Sigma}_\mathbf{s}] + \text{tr}[\mathbf{A}\mathbf{W}(\mathbf{H}\boldsymbol{\Sigma}_\mathbf{s}\mathbf{H}^T + \sigma_\nu^2\mathbf{I}_N)\mathbf{W}^T\mathbf{A}^T] + \sigma_\delta^2\text{tr}[\mathbf{A}\mathbf{A}^T] \tag{4}$$

with $\mathcal{E} = \text{tr}\langle\boldsymbol{\epsilon}\boldsymbol{\epsilon}^T\rangle$ by definition, $\langle\cdot\rangle$ the average over samples, and $\boldsymbol{\Sigma}_\mathbf{s}$ the covariance matrix of the image signal $\mathbf{s}$. The problem is to find $\mathbf{W}$ and $\mathbf{A}$ that minimize $\mathcal{E}$.

To model limited neural capacity, the representation $\mathbf{r}$ must have limited SNR. This constraint is equivalent to fixing the variance of filter output $\langle\mathbf{w}_j^T\mathbf{x}\rangle = \sigma_u^2$, where $\mathbf{w}_j$ is the $j$-th row of $\mathbf{W}$ (here we assume all neurons have the same capacity). It is expressed in the matrix form as

$$\text{diag}[\mathbf{W}\boldsymbol{\Sigma}_\mathbf{x}\mathbf{W}^T] = \sigma_u^2\mathbf{1}_M \tag{5}$$

where $\boldsymbol{\Sigma}_\mathbf{x} = \mathbf{H}\boldsymbol{\Sigma}_\mathbf{s}\mathbf{H}^T + \sigma_\nu^2\mathbf{I}_N$ is the covariance of the observation. It can further be simplified to

$$\text{diag}[\mathbf{V}\mathbf{V}^T] = \mathbf{1}_M, \tag{6}$$
$$\mathbf{W} = \sigma_u\mathbf{V}\mathbf{S}_\mathbf{x}^{-1}\mathbf{E}^T, \tag{7}$$

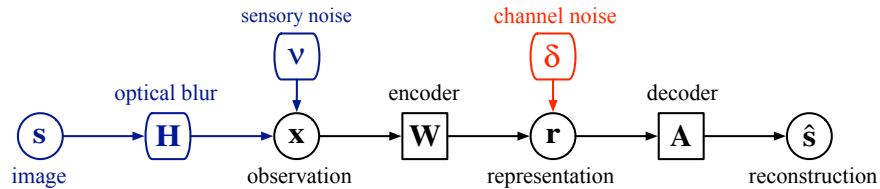

Figure 2: The model diagram. If there is no degradation of the image ($\mathbf{H} = \mathbf{I}$ and $\sigma_\nu^2 = 0$), the model is reduced to the original robust coding model [2]. If channel noise is zero as well ($\sigma_\delta^2 = 0$), it boils down to conventional block coding such as PCA, ICA, or wavelet transforms.

where $\mathbf{S_x} = \mathrm{diag}(\sqrt{\alpha_1\lambda_1 + \sigma_\nu^2}, \cdots, \sqrt{\alpha_N\lambda_N + \sigma_\nu^2})$ (the square root of $\mathbf{\Sigma_x}$'s eigenvalues), $\sqrt{\alpha_k}$ and $\lambda_k$ are respectively the eigenvalues of $\mathbf{H}$ and $\mathbf{\Sigma_s}$, and the columns of $\mathbf{E}$ are their common eigenvectors[1]. Note that $\sqrt{\alpha_k}$ defines the modulation transfer function of the optical blur $\mathbf{H}$, i.e., the attenuation of the amplitude of the signal along the $k$-th eigenvector.

Now, the problem is to find $\mathbf{V}$ and $\mathbf{A}$ that minimize $\mathcal{E}$. The optimal $\mathbf{A}$ should satisfy $\partial\mathcal{E}/\partial\mathbf{A} = \mathbf{O}$, which yields

$$\mathbf{A} = \mathbf{\Sigma_s}\mathbf{H}^T\mathbf{W}^T[\mathbf{W}(\mathbf{H}\mathbf{\Sigma_s}\mathbf{H}^T + \sigma_\nu^2\mathbf{I}_N)\mathbf{W}^T + \sigma_\delta^2\mathbf{I}_M]^{-1} \tag{8}$$

$$= \frac{\gamma^2}{\sigma_u}\mathbf{E}\mathbf{S_s}\mathbf{P}[\mathbf{I}_N + \gamma^2\mathbf{V}^T\mathbf{V}]^{-1}\mathbf{V}^T \tag{9}$$

where $\gamma^2 = \sigma_u^2/\sigma_\delta^2$ (neural SNR), $\mathbf{S_s} = \mathrm{diag}(\sqrt{\lambda_1}, \cdots, \sqrt{\lambda_N})$, $\mathbf{P} = \mathrm{diag}(\sqrt{\phi_1}, \cdots, \sqrt{\phi_N})$, and $\phi_k = \alpha_k\lambda_k/(\alpha_k\lambda_k + \sigma_\nu^2)$ (the power ratio between the attenuated signal and that signal plus sensory noise; as we will see below, $\phi_k$ characterizes the generalized solutions of robust coding, and if there is neither sensory noise nor optical blur, $\phi_k$ becomes 1 that reduces the solutions of the current model to those in the original robust coding model [2]). This implies that the optimal $\mathbf{A}$ is determined once the optimal $\mathbf{V}$ is found.

With eqn. 7 and 9, $\mathcal{E}$ becomes

$$\mathcal{E} = \sum_{k=1}^{N}\lambda_k(1 - \phi_k) + \mathrm{tr}[\mathbf{S_s}^2\mathbf{P}^2(\mathbf{I}_N + \gamma^2\mathbf{V}^T\mathbf{V})^{-1}]. \tag{10}$$

Finally, the problem is reduced to finding $\mathbf{V}$ that minimizes eqn. 10.

**Solutions for 2-D data**

In this section we present the explicit characterization of the optimal solutions for two-dimensional data. It entails under-complete, complete, and over-complete representations, and provides precise insights into the numerical solutions for the high-dimensional image data (Section 3). This is a generalization of the analysis in [2] with the addition of optical blur and additive sensory noise.

From eqn. 6 we can parameterize $\mathbf{V}$ with

$$\mathbf{V} = \begin{pmatrix} \cos\theta_1 & \sin\theta_1 \\ \vdots & \vdots \\ \cos\theta_M & \sin\theta_M \end{pmatrix} \tag{11}$$

where $\theta_j \in [0, 2\pi)$, $j = 1, \cdots, M$, which yields

$$\mathcal{E} = \sum_{k=1}^{2}\lambda_k(1 - \phi_k) + \frac{(\psi_1 + \psi_2)\left(\frac{M}{2}\gamma^2 + 1\right) - \frac{\gamma^2}{2}(\psi_1 - \psi_2)\,\mathrm{Re}(Z)}{\left(\frac{M}{2}\gamma^2 + 1\right)^2 - \frac{1}{4}\gamma^4|Z|^2}, \tag{12}$$

with $\psi_k \equiv \phi_k\lambda_k$ and $Z \equiv \sum_j(\cos 2\theta_j + i\sin 2\theta_j)$. In the following we analyze the cases when $\psi_1 = \psi_2$ and when $\psi_1 \neq \psi_2$. Without loss of generality we consider $\psi_1 > \psi_2$ for the latter case. (In the previous analysis of robust coding [2], these cases depend only on the ratio between $\lambda_1$ and $\lambda_2$, i.e., the isotropy of the data. In the current, general model, these also depend on the isotropy of the optical blur ($\alpha_1$ and $\alpha_2$) and the variance of sensory noise ($\sigma_\nu^2$), and no simple meaning is attached to the individual cases.)

1). If $\psi_1 = \psi_2$ ($\equiv \psi$): $\mathcal{E}$ in eqn. 10 becomes

$$\mathcal{E} = \sum_{k=1}^{2}\lambda_k(1 - \phi_k) + \frac{2\psi\left(\frac{M}{2}\gamma^2 + 1\right)}{\left(\frac{M}{2}\gamma^2 + 1\right)^2 - \frac{1}{4}\gamma^4|Z|^2}. \tag{13}$$

Therefore, $\mathcal{E}$ is minimized when $|Z|^2$ is minimized.

1-a). If $M = 1$ (single neuron case): By definition $|Z|^2 = 1$, implying that $\mathcal{E}$ is constant for any $\theta_1$,

$$\mathcal{E} \;=\; \left[\lambda_1(1-\phi_1) + \frac{\psi_1}{\gamma^2+1}\right] + \lambda_2 \;=\; \left[\lambda_2(1-\phi_2) + \frac{\psi_2}{\gamma^2+1}\right] + \lambda_1, \tag{14}$$

$$\mathbf{W} \;=\; \sigma_u \left(\begin{array}{cc} \cos\theta_1 & \sin\theta_1 \end{array}\right) \left(\begin{array}{cc} 1/\sqrt{\alpha_1\lambda_1 + \sigma_\nu^2} & 0 \\ 0 & 1/\sqrt{\alpha_2\lambda_2 + \sigma_\nu^2} \end{array}\right) \mathbf{E}^T. \tag{15}$$

Because there is only one neuron, only one direction in the two dimensional space can be reconstructed, and eqn. 15 implies that any direction can be equally good. The first equality in eqn. 14 can be interpreted as the case when $\mathbf{W}$ represents the direction along the first eigenvector, and consequently, the whole data variance along the second eigenvector $\lambda_2$ is left in the error $\mathcal{E}$.

1-b). If $M \geq 2$ (multiple neuron case): There always exists $Z$ that satisfies $|Z| = 0$ if $M \geq 2$, with which $\mathcal{E}$ is minimized [2]. Accordingly,

$$\mathcal{E} \;=\; \sum_{k=1}^{2}\left[\lambda_k(1-\phi_k) + \frac{\psi_k}{\frac{M}{2}\gamma^2+1}\right], \tag{16}$$

$$\mathbf{W} \;=\; \sigma_u \mathbf{V} \left(\begin{array}{cc} 1/\sqrt{\alpha_1\lambda_1 + \sigma_\nu^2} & 0 \\ 0 & 1/\sqrt{\alpha_2\lambda_2 + \sigma_\nu^2} \end{array}\right) \mathbf{E}^T, \tag{17}$$

where $\mathbf{V}$ is arbitrary as long as it satisfies $|Z| = 0$. Note that $\mathbf{W}$ takes the same form as in $M = 1$ except that there are more than two neurons. Also, eqn. 16 shares the second term with eqn. 14 except that the SNR of the representation $\gamma^2$ is multiplied by $M/2$. It implies that having $n$ times the neurons is equivalent to increasing the representation SNR by the factor of $n$ (this relation generally holds in the multiple neuron cases below).

2). If $\psi_1 > \psi_2$: Eqn. 12 is minimized when $Z = \mathrm{Re}(Z) \geq 0$ for a fixed value of $|Z|^2$. Therefore, the problem is reduced to seeking a real value $Z = y \in [0, M]$ that minimizes

$$\mathcal{E} = \sum_{k=1}^{2} \lambda_k(1-\phi_k) + \frac{(\psi_1+\psi_2)\left(\frac{M}{2}\gamma^2+1\right) - \frac{\gamma^2}{2}(\psi_1-\psi_2)y}{\left(\frac{M}{2}\gamma^2+1\right)^2 - \frac{1}{4}\gamma^4 y^2}. \tag{18}$$

2-a). If $M = 1$ (single neuron case): $Z = \mathrm{Re}(Z)$ holds iff $\theta_1 = 0$. Accordingly,

$$\mathcal{E} \;=\; \left[\lambda_1(1-\phi_1) + \frac{\psi_1}{\gamma^2+1}\right] + \lambda_2, \tag{19}$$

$$\mathbf{W} \;=\; \frac{\sigma_u}{\sqrt{\alpha_1\lambda_1 + \sigma_\nu^2}}\mathbf{e}_1^T. \tag{20}$$

These take the same form as in the case of $\psi_1 = \psi_2$ and $M = 1$ (eqn. 14-15) except that the direction of the representation is specified along the first eigenvector $\mathbf{e}_1$, indicating that all the representational resources (namely, one neuron) are devoted to the largest data variance direction.

2-b). If $M \geq 2$ (multiple neuron case): From eqn. 18, the necessary condition for the minimun $d\mathcal{E}/dy = 0$ yields

$$\left[\frac{\sqrt{\psi_1}-\sqrt{\psi_2}}{\sqrt{\psi_1}+\sqrt{\psi_2}}\left(M + \frac{2}{\gamma^2}\right) - y\right]\left[\frac{\sqrt{\psi_1}+\sqrt{\psi_2}}{\sqrt{\psi_1}-\sqrt{\psi_2}}\left(M + \frac{2}{\gamma^2}\right) - y\right] = 0. \tag{21}$$

The existence of a root $y$ in the domain $[0, M]$ depends on how $\gamma^2$ compares to the next quantity, which is a generalized form of the critical point of neural precision [2]:

$$\gamma_c^2 = \frac{1}{M}\left[\sqrt{\frac{\psi_1}{\psi_2}} - 1\right]. \tag{22}$$

2-b-i). If $\gamma^2 < \gamma_c^2$: $d\mathcal{E}/dy = 0$ does not have a root within the domain. Since $d\mathcal{E}/dy$ is always negative, $\mathcal{E}$ is minimized when $y = M$. Accordingly,

$$\mathcal{E} \;=\; \left[\lambda_1(1-\phi_1) + \frac{\psi_1}{M\gamma^2+1}\right] + \lambda_2, \tag{23}$$

$$\mathbf{W} \;=\; \frac{\sigma_u}{\sqrt{\alpha_1\lambda_1 + \sigma_\nu^2}}\mathbf{1}_M\mathbf{e}_1^T. \tag{24}$$

These solutions are the same as in $M = 1$ (eqn. 19-20) except that the neural SNR $\gamma^2$ is multiplied by $M$ to yield smaller MSE.

2-b-ii). If $\gamma^2 \geq \gamma_c^2$: Eqn. 21 has a root within $[0, M]$,

$$y = \frac{\sqrt{\psi_1} - \sqrt{\psi_2}}{\sqrt{\psi_1} + \sqrt{\psi_2}} \left( \frac{2}{\gamma^2} + M \right), \tag{25}$$

with $y = M$ if $\gamma^2 = \gamma_c^2$. The optimal solutions are

$$\mathcal{E} = \sum_{k=1}^{2} \lambda_k (1 - \phi_k) + \frac{1}{\frac{M}{2}\gamma^2 + 1} \frac{(\sqrt{\psi_1} + \sqrt{\psi_2})^2}{2}, \tag{26}$$

$$\mathbf{W} = \sigma_u \mathbf{V} \begin{pmatrix} 1/\sqrt{\alpha_1 \lambda_1 + \sigma_\nu^2} & 0 \\ 0 & 1/\sqrt{\alpha_2 \lambda_2 + \sigma_\nu^2} \end{pmatrix} \mathbf{E}^T, \tag{27}$$

where $\mathbf{V}$ is arbitrary up to satisfying eqn. 25.

In Fig. 3 we illustrate some examples of explicit solutions for 2-D data with two neurons. The general strategy of the proposed model is to represent the principal axis of the signal $\mathbf{s}$ more accurately as the signal is more degraded (by optical blur and/or sensory noise). Specifically, the two neurons come to represent the identical dimension when the degradation is sufficiently large.

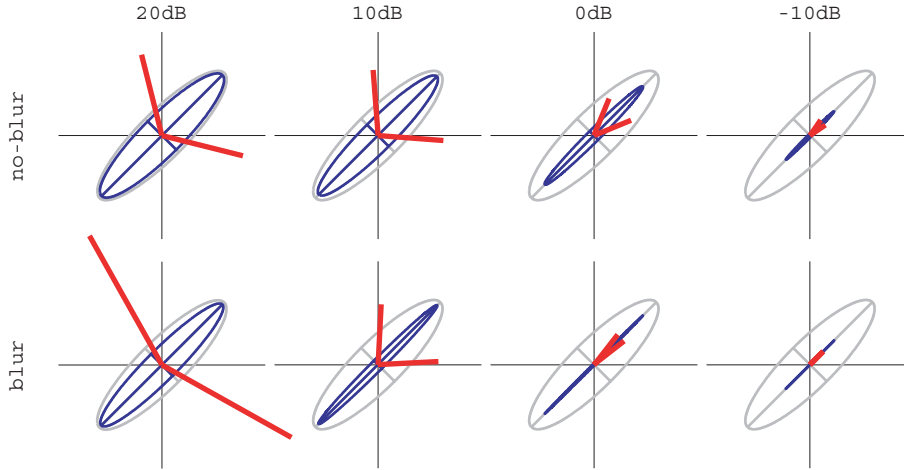

Figure 3: Sensory noise changes the optimal linear filter. The gray (outside) and blue (inside) contours show the variance of the target and reconstructed signal, respectively, and the red (thick) bars the optimal linear filters when there are two neurons. The SNR of the observation is varied from 20 to $-10$ dB (column-wise). The bottom row is the case where the power of the signal's minor component is attenuated as in the optical blur (i.e., low pass filtering): $(\alpha_1, \alpha_2) = (1, 0.1)$; while the top is without the blur: $(\alpha_1, \alpha_2) = (1, 1)$. The neural SNR is fixed at 10 dB.

## 3  Optimal receptive field populations

We applied the proposed model to a natural images data set [11] to obtain the theoretically optimal population coding for mRGCs. The optimal solutions were derived under the following biological constraints on the observation, or the photoreceptor response, $\mathbf{x}$ (Fig. 2). To model the retinal images at different retinal eccentricities, we used modulation transfer functions of the human eye [3] and cone photoreceptor densities of the human retina [1] (Fig. 1). The retinal image is further corrupted by the additive Gaussian noise to model the photon transduction noise by which the SNR of the observartion becomes smaller under dimmer illumination level [4]. This yields the observation at different retinal eccentricities. In the following, we present the optimal solutions for the fovea (where the most accurate visual information is represented while the receptive field characteristics are difficult to measure experimentally) and those at 40 degrees retinal eccentricity (where we can compare the model to recent physiological measurements in the primate retina [12]).

The information capacity of neural representations is limited by both the number of neurons and the precision of neural codes. The ratio of cone photoreceptors to mRGCs in the human retina is $1 : 2$ at the fovea and $23 : 2$ at 40 degrees [13]. We did not model neural rectification (separate on and off channels) and thus assumed the effective cell ratios as $1 : 1$ and $23 : 1$, respectively. We also fixed the neural SNR at 10 dB, equivalent to assuming $\sim 1.7$ bits coding precision as in real neurons [14].

The optimal $\mathbf{W}$ can be derived with the gradient descent on $\mathcal{E}$, and $\mathbf{A}$ can be derived from $\mathbf{W}$ using eqn. 8. As explained in Section 2, the solution must satisfy the variance constraint (eqn. 6). We formulate this as a constrained optimization problem [15]. The update rule for $\mathbf{W}$ is given by

$$\Delta \mathbf{W} \propto -\mathbf{A}^T(\mathbf{AWH} - \mathbf{I}_N)\mathbf{\Sigma_s}\mathbf{H}^T - \sigma_\nu^2 \mathbf{A}^T\mathbf{AW} - \kappa \operatorname{diag}\left(\frac{\ln[\operatorname{diag}(\mathbf{W\Sigma_x}\mathbf{W}^T)/\sigma_u^2]}{\operatorname{diag}(\mathbf{W\Sigma_x}\mathbf{W}^T)}\right)\mathbf{W\Sigma_x}, \quad (28)$$

where $\kappa$ is a positive constant that controls the strength of the variance constraint. Our initial results indicated that the optimal solutions are not unique and these solutions are equivalent in terms of MSE. We then imposed an additional neural resource constraint that penalizes the spatial extent of a receptive field: the constraint for the $k$-th neuron is defined by $\sum_j |W_{kj}|(\rho\, d_{kj}^2 + 1)$ where $d_{kj}$ is the spatial distance between the $j$-th weight and the center of mass of all weights, and $\rho$ is a positive constant defining the strength of the spatial constraint. This assumption is consistent with the spatially restricted computation in the retina. If $\rho = 0$, it imposes sparse weights [16], though not necessarily spatially localized. In our simulations we fixed $\rho = 0.5$.

For the fovea, we examined $15 \times 15$ pixel image patches sampled from a large set of natural images, where each pixel corresponds to a cone photoreceptor. Since the cell ratio is assumed to be $1 : 1$, there were 225 model neurons in the population. As shown in Fig. 4, the optimal filters show concentric center-surround organization that is well fit with a difference-of-Gaussian function (which is one major characteristic of mRGCs). The precise organization of the model receptive field changes according to the SNR of the observation: as the SNR decreases, the surround inhibition gradually disappears and the center becomes larger, which serves to remove sensory noise by averaging. As a population, this yields a significant overlap among adjacent receptive fields. In terms of spatial-frequency, this change corresponds to a shift from band-pass to low-pass filtering, which is consistent with psychophysical measurements of the human and the macaque [17].

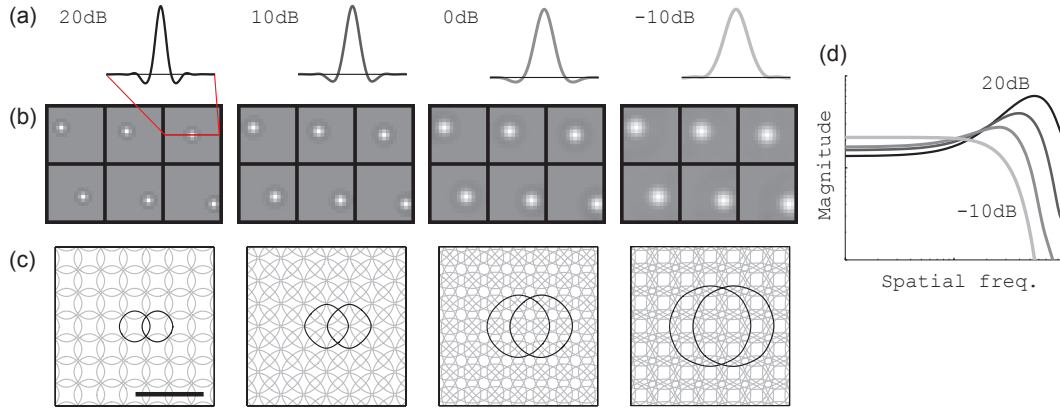

Figure 4: The model receptive fields at the fovea under different SNRs of the observation. (a) A cross-section of the two-dimensional receptive field. (b) Six examples of receptive fields. (c) The tiling of a population of receptive fields in the visual field. The ellipses show the contour of receptive fields at half the maximum. One pair of adjacent filters are highlighted for clarity. The scale bar indicates an interval of three photoreceptors. (d) Spatial-frequency profiles (modulation transfer functions) of the receptive fields at different SNRs.

For 40 degrees retinal eccentricity, we examined $35 \times 35$ photoreceptor array that are projected to 53 model neurons (so that the cell ratio is $23 : 1$). The general trend of the results is the same as in the fovea except that the receptive fields are much larger. This allows the fewer neurons in the population to completely tile the visual field. Furthermore, the change of the receptive field with the sensory noise level is not as significant as that predicted for the fovea, suggesting that the SNR is a

less significant factor when neural number is severely limited. We also note that the elliptical shape of the extent of the receptive fields matches experimental observations [12].

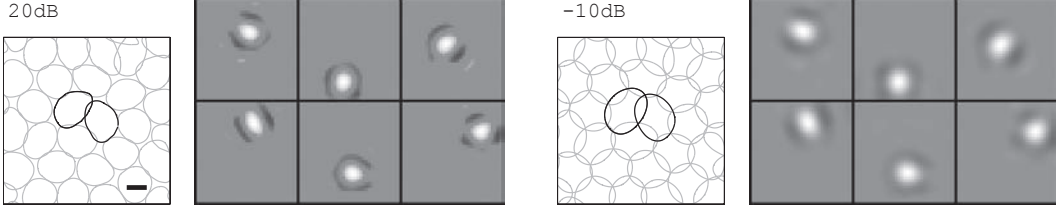

Figure 5: The theoretically-derived receptive fields for 40 degrees of the retinal eccentricity. Captions as in Fig. 4.

Finally, we demonstrate the performance of de-blurring, de-noising, and information preservation by these receptive fields (Fig. 6). The original image is well recovered in spite of both the noisy representation (10% of the code's variation is noise because of the 10 dB precision) and the noisy, degraded observation. Note that the 40 degrees eccentricity is subject to an additional, significant dimensionality reduction, which is why the reconstruction error (e.g., 34.8% at 20 dB) can be greater than the distortion in the observation (30.5%).

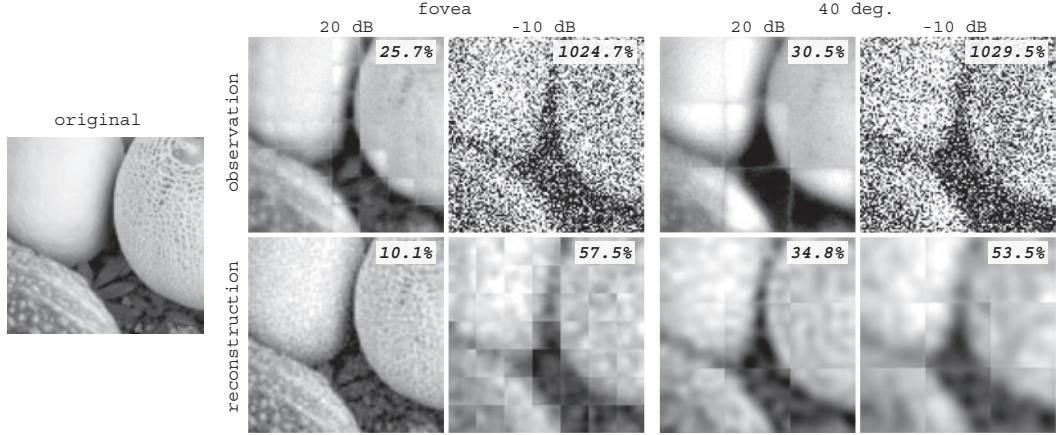

Figure 6: Reconstruction example. For both the fovea and 40 degrees retinal eccentricity, two sensory noise conditions are shown (20 and $-10$ dB). The percentages indicate the average distortion in the observation or the reconstruction error, respectively, over 60,000 samples. The blocking effect is caused by the implementation of the optical blur on each image patch using a matrix $\mathbf{H}$ instead of convolving the whole image.

## 4  Discussion

The proposed model is a generalization of the robust coding model [2] and allows a complete characterization of the optimal representation as a function of both image degradation (optical blur and additive sensory noise) and limited neural capacity (neural precision and population size). If there is no sensory noise $\sigma_\nu^2 = 0$ and no optical blur $\mathbf{H} = \mathbf{I}_N$, then $\phi_k = 1$ for all $k$, which reduces all the optimal solutions above to those reported in [2].

The proposed model may also be viewed as a generalization of the Wiener filter: if there is no channel noise $\sigma_\delta^2 = 0$ and the cell ratio is $1 : 1$, and by assuming $\mathbf{A} \equiv \mathbf{I}_N$ without loss of generality, the problem is reformulated as finding $\mathbf{W} \in \mathbb{R}^{N \times N}$ that provides the best estimate of the original signal $\hat{\mathbf{s}} = \mathbf{W}(\mathbf{Hs} + \boldsymbol{\nu})$ in terms of the MSE. The optimal solution is given by the Wiener filter:

$$\mathbf{W} = \boldsymbol{\Sigma}_{\mathbf{s}}\mathbf{H}^T[\mathbf{H}\boldsymbol{\Sigma}_{\mathbf{s}}\mathbf{H}^T + \sigma_\nu^2\mathbf{I}_N]^{-1} = \mathbf{E}\,\mathrm{diag}\left[\frac{\sqrt{\alpha_1}\lambda_1}{\alpha_1\lambda_1 + \sigma_\nu^2}, \cdots, \frac{\sqrt{\alpha_N}\lambda_N}{\alpha_N\lambda_N + \sigma_\nu^2}\right]\mathbf{E}^T, \quad (29)$$

$$\mathcal{E} \;=\; \text{tr}[\boldsymbol{\Sigma_s}] - \text{tr}[\mathbf{WH\Sigma_s}] \;=\; \sum_{k=1}^{N} \lambda_k (1 - \phi_k), \tag{30}$$

(note that the diagonal matrix in eqn. 29 corresponds to the Wiener filter formula in the frequency domain [5]). This also implies that the Wiener filter is optimal only in the limiting case in our setting.

Here, we have treated the model primarily as a theory of retinal coding, but its generality would allow it to be applied to a wide range of problems in signal processing. We should also note several limitations. The model assumes Gaussian signal structure. Modeling non-Gaussian signal distributions might account for coding efficiency constraints on the retinal population. The model is linear, but the framework allows for the incorporation of non-linear encoding and decoding methods, at the expense of analytic tractability.

There have been earlier approaches to theoretically characterizing the retinal code [4, 6, 7, 8]. Our approach differs from these in several respects. First, it is not restricted to the so-called complete representation ($M = N$) and can predict properties of mRGCs at any retinal eccentricity. Second, we do not assume a single, translation invariant filter and can derive the optimal receptive fields for a neural population. Third, we accurately model optical blur, retinal sampling, cell ratio, and neural precision. Finally, we assumed that, as in [4, 8], the objective of the retinal coding is to form the neural code that yields the minimum MSE with linear decoding, while others assumed it to form the neural code that maximally preserves information about signal [6, 7]. To the best of our knowledge, we don't know a priori which objective should be appropriate for the retinal coding. As suggested earlier [8], this issue could be resolved by comparing different theoretical predictions to physiological data.

## Footnotes

[1]The eigenvectors of $\mathbf{\Sigma_s}$ and $\mathbf{H}$ are both Fourier basis functions because we assume that $\mathbf{s}$ are natural images [9] and $\mathbf{H}$ is a circulant matrix [10].

# References

[1] R. W. Rodieck. *The First Steps in Seeing*. Sinauer, MA, 1998.

[2] E. Doi, D. C. Balcan, and M. S. Lewicki. A theoretical analysis of robust coding over noisy overcomplete channels. In *Advances in Neural Information Processing Systems*, volume 18. MIT Press, 2006.

[3] R. Navarro, P. Artal, and D. R. Williams. Modulation transfer of the human eye as a function of retinal eccentricity. *Journal of Optical Society of America A*, 10:201–212, 1993.

[4] M. V. Srinivasan, S. B. Laughlin, and A. Dubs. Predictive coding: a fresh view of inhibition in the retina. *Proc. R. Soc. Lond. B*, 216:427–459, 1982.

[5] R. C. Gonzalez and R. E. Woods. *Digital image processing*. Prentice Hall, 2nd edition, 2002.

[6] J. J. Atick and A. N. Redlich. Towards a theory of early visual processing. *Neural Computation*, 2:308–320, 1990.

[7] J. H. van Hateren. Theoretical predictions of spatiotemporal receptive fields of fly LMCs, and experimental validation. *J. Comp. Physiol. A*, 171:157–170, 1992.

[8] D. L. Ruderman. Designing receptive fields for highest fidelity. *Network*, 5:147–155, 1994.

[9] D. J. Field. Relations between the statistics of natural images and the response properties of cortical cells. *J. Opt. Soc. Am. A*, 4:2379–2394, 1987.

[10] R. M. Gray. Toeplitz and circulant matrices: A review. *Foundations and Trends in Communications and Information Theory*, 2:155–239, 2006.

[11] E. Doi, T. Inui, T.-W. Lee, T. Wachtler, and T. J. Sejnowski. Spatiochromatic receptive field properties derived from information-theoretic analyses of cone mosaic responses to natural scenes. *Neural Computation*, 15:397–417, 2003.

[12] E. S. Frechette, A. Sher, M. I. Grivich, D. Petrusca, A. M. Litke, and E. J. Chichilnisky. Fidelity of the ensemble code for visual motion in primate retina. *Journal of Neurophysiology*, 94:119–135, 2005.

[13] C. A. Curcio and K. A. Allen. Topography of ganglion cells in human retina. *Journal of Comparative Neurology*, 300:5–25, 1990.

[14] A. Borst and F. E. Theunissen. Information theory and neural coding. *Nature Neuroscience*, 2:947–957, 1999.

[15] E. Doi and M. S. Lewicki. Sparse coding of natural images using an overcomplete set of limited capacity units. In *Advances in Neural Information Processing Systems*, volume 17. MIT Press, 2005.

[16] B. T. Vincent and R. J. Baddeley. Synaptic energy efficiency in retinal processing. *Vision Research*, 43:1283–1290, 2003.

[17] R. L. De Valois, H. Morgan, and D. M. Snodderly. Psychophysical studies of monkey vision - III. Spatial luminance contrast sensitivity test of macaque and human observers. *Vision Research*, 14:75–81, 1974.
